# Soft Clustering on Graphs

**Kai Yu**[1]**, Shipeng Yu**[2]**, Volker Tresp**[1]
[1]Siemens AG, Corporate Technology
[2]Institute for Computer Science, University of Munich
kai.yu@siemens.com, volker.tresp@siemens.com
spyu@dbs.informatik.uni-muenchen.de

## Abstract

We propose a simple clustering framework on graphs encoding pairwise data similarities. Unlike usual similarity-based methods, the approach softly assigns data to clusters in a probabilistic way. More importantly, a *hierarchical clustering* is naturally derived in this framework to gradually merge lower-level clusters into higher-level ones. A random walk analysis indicates that the algorithm exposes clustering structures in various *resolutions*, i.e., a higher level statistically models a longer-term diffusion on graphs and thus discovers a more *global* clustering structure. Finally we provide very encouraging experimental results.

## 1 Introduction

Clustering has been widely applied in data analysis to group similar objects. Many algorithms are either similarity-based or model-based. In general, the former (e.g., normalized cut [5]) requires no assumption on data densities but simply a similarity function, and usually partitions data exclusively into clusters. In contrast, model-based methods apply mixture models to fit data distributions and assign data to clusters (i.e. mixture components) probabilistically. This *soft* clustering is often desired, as it encodes uncertainties on data-to-cluster assignments. However, their density assumptions can sometimes be restrictive, e.g. clusters have to be Gaussian-like in Gaussian mixture models (GMMs).

In contrast to flat clustering, *hierarchical clustering* makes intuitive senses by forming a tree of clusters. Despite of its wide applications, the technique is usually achieved by heuristics (e.g., single link) and lacks theoretical backup. Only a few principled algorithms exist so far, where a Gaussian or a sphere-shape assumption is often made [3, 1, 2].

This paper suggests a novel graph-factorization clustering (GFC) framework that employs data's affinities and meanwhile partitions data probabilistically. A hierarchical clustering algorithm (HGFC) is further derived by merging lower-level clusters into higher-level ones. Analysis based on graph random walks suggests that our clustering method models data affinities as *empirical transitions* generated by a mixture of latent factors. This view significantly differs from conventional model-based clustering since here the mixture model is not directly for data objects but for their relations. Clusters with arbitrary shapes can be modeled by our method since only pairwise similarities are considered. Interestingly, we prove that the higher-level clusters are associated with longer-term diffusive transitions on the graph, amounting to smoother and more global similarity functions on the data mani-

fold. Therefore, the cluster hierarchy exposes the observed affinity structure gradually in different *resolutions*, which is somehow similar to the *wavelet* method that analyzes signals in different bandwidths. To the best of our knowledge, this property has never been considered by other agglomerative hierarchical clustering algorithms (e.g., see [3]).

The paper is organized as follows. In the following section we describe a clustering algorithm based on similarity graphs. In Sec. 3 we generalize the algorithm to hierarchical clustering, followed by a discussion from the random walk point of view in Sec. 4. Finally we present the experimental results in Sec. 5 and conclude the paper in Sec. 6.

## 2  Graph-factorization clustering (GFC)

Data similarity relations can be conveniently encoded by a graph, where vertices denote data objects and adjacency weights represent data similarities. This section introduces *graph factorization clustering*, which is a probabilistic partition of graph vertices. Formally, let $G(\mathbf{V}, \mathbf{E})$ be a weighted undirected graph with vertices $\mathbf{V} = \{v_i\}_{i=1}^n$ and edges $\mathbf{E} \subseteq \{(v_i, v_j)\}$. Let $\mathbf{W} = \{w_{ij}\}$ be the adjacency matrix, where $w_{ij} = w_{ji}$, $w_{ij} > 0$ if $(v_i, v_j) \in \mathbf{E}$ and $w_{ij} = 0$ otherwise. For instances, $w_{ij}$ can be computed by the RBF similarity function based on the features of objects $i$ and $j$, or by a binary indicator (0 or 1) of the $k$-nearest neighbor affinity.

### 2.1  Bipartite graphs

Before presenting the main idea, it is necessary to introduce *bipartite graphs*. Let $K(\mathbf{V}, \mathbf{U}, \mathbf{F})$ be the bipartite graph (e.g., Fig. 1–(b)), where $\mathbf{V} = \{v_i\}_{i=1}^n$ and $\mathbf{U} = \{u_p\}_{p=1}^m$ are the two disjoint vertex sets and $\mathbf{F}$ contains all the edges connecting $\mathbf{V}$ and $\mathbf{U}$. Let $\mathbf{B} = \{b_{ip}\}$ denote the $n \times m$ adjacency matrix with $b_{ip} \geq 0$ being the weight for edge $[v_i, u_p]$. The bipartite graph $K$ induces a similarity between $v_1$ and $v_j$ [6]

$$w_{ij} = \sum_{p=1}^m \frac{b_{ip} b_{jp}}{\lambda_p} = \left( \mathbf{B} \boldsymbol{\Lambda}^{-1} \mathbf{B}^\top \right)_{ij}, \quad \boldsymbol{\Lambda} = \mathrm{diag}(\lambda_1, \ldots, \lambda_m) \tag{1}$$

where $\lambda_p = \sum_{i=1}^n b_{ip}$ denotes the degree of vertex $u_p \in \mathbf{U}$. We can interpret Eq. (1) from the perspective of *Markov random walks* on graphs. $w_{ij}$ is essentially a quantity proportional to the stationary probability of direct transitions between $v_i$ and $v_j$, denoted by $p(v_i, v_j)$. Without loss of generality, we normalize $\mathbf{W}$ to ensure $\sum_{ij} w_{ij} = 1$ and $w_{ij} = p(v_i, v_j)$. For a bipartite graph $K(\mathbf{V}, \mathbf{U}, \mathbf{F})$, there is no direct links between vertices in $\mathbf{V}$, and all the paths from $v_i$ to $v_j$ must go through vertices in $\mathbf{U}$. This indicates

$$p(v_i, v_j) = p(v_i) p(v_j | v_i) = d_i \sum_p p(u_p | v_i) p(v_j | u_p) = \sum_p \frac{p(v_i, u_p) p(u_p, v_j)}{\lambda_p},$$

where $p(v_j | v_i)$ is the conditional transition probability from $v_i$ to $v_j$, and $d_i = p(v_i)$ the degree of $v_i$. This directly leads to Eq. (1) with $b_{ip} = p(v_i, u_p)$.

### 2.2  Graph factorization by bipartite graph construction

For a bipartite graph $K$, $p(u_p | v_i) = b_{ip}/d_i$ tells the conditional probability of transitions from $v_i$ to $u_p$. If the size of $\mathbf{U}$ is smaller than that of $\mathbf{V}$, namely $m < n$, then $p(u_p | v_i)$ indicates how likely data point $i$ belongs to vertex $p$. This property suggests that one can construct a bipartite graph $K(\mathbf{V}, \mathbf{U}, \mathbf{F})$ to approximate a given $G(\mathbf{V}, \mathbf{E})$, and then obtain a soft clustering structure, where $\mathbf{U}$ corresponds to clusters (see Fig. 1–(a) (b)).

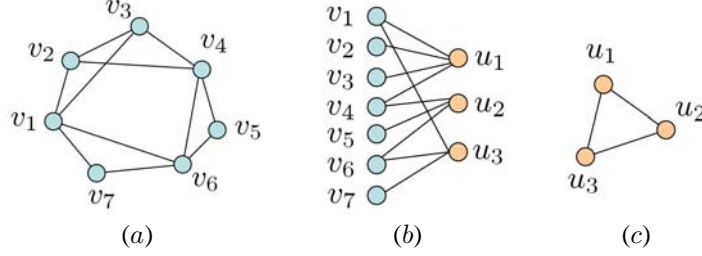

Figure 1: (*a*) The original graph representing data affinities; (*b*) The bipartite graph representing data-to-cluster relations; (*c*) The induced cluster affinities.

Eq. (1) suggests that this approximation can be done by minimizing $\ell(\mathbf{W}, \mathbf{B}\mathbf{\Lambda}^{-1}\mathbf{B}^\top)$, given a distance $\ell(\cdot, \cdot)$ between two adjacency matrices. To make the problem easy to solve, we remove the coupling between $\mathbf{B}$ and $\mathbf{\Lambda}$ via $\mathbf{H} = \mathbf{B}\mathbf{\Lambda}^{-1}$ and then have

$$\min_{\mathbf{H}, \mathbf{\Lambda}} \ell\left(\mathbf{W}, \mathbf{H}\mathbf{\Lambda}\mathbf{H}^\top\right), \qquad \text{s. t. } \sum_{i=1}^{n} h_{ip} = 1, \mathbf{H} \in \mathbb{R}_+^{n \times m}, \mathbf{\Lambda} \in \mathbb{D}_+^{m \times m}, \qquad (2)$$

where $\mathbb{D}_+^{m \times m}$ denotes the set of $m \times m$ diagonal matrices with positive diagonal entries. This problem is a symmetric variant of non-negative matrix factorization [4]. In this paper we focus on the *divergence distance* between matrices. The following theorem suggests an alternating optimization approach to find a local minimum:

**Theorem 2.1.** *For divergence distance* $\ell(\mathbf{X}, \mathbf{Y}) = \sum_{ij}(x_{ij} \log \frac{x_{ij}}{y_{ij}} - x_{ij} + y_{ij})$, *the cost function in Eq. (2) is non-increasing under the update rule ( $\tilde{\cdot}$ denote updated quantities)*

$$\tilde{h}_{ip} \propto h_{ip} \sum_j \frac{w_{ij}}{(\mathbf{H}\mathbf{\Lambda}\mathbf{H}^\top)_{ij}} \lambda_p h_{jp}, \quad \text{normalize s.t. } \sum_i \tilde{h}_{ip} = 1; \qquad (3)$$

$$\tilde{\lambda}_p \propto \lambda_p \sum_{ij} \frac{w_{ij}}{(\mathbf{H}\mathbf{\Lambda}\mathbf{H}^\top)_{ij}} h_{ip} h_{jp}, \quad \text{normalize s.t. } \sum_p \tilde{\lambda}_p = \sum_{ij} w_{ij}. \qquad (4)$$

*The distance is invariant under the update if and only if* $\mathbf{H}$ *and* $\mathbf{\Lambda}$ *are at a stationary point.*

See Appendix for all the proofs in this paper. Similar to GMM, $p(u_p|v_i) = b_{ip}/\sum_q b_{iq}$ is the *soft* probabilistic assignment of vertex $v_i$ to cluster $u_p$. The method can be seen as a counterpart of mixture models on graphs. The time complexity is $\mathcal{O}(m^2 N)$ with $N$ being the number of nonzero entries in $\mathbf{W}$. This can be very efficient if $\mathbf{W}$ is sparse (e.g., for $k$-nearest neighbor graph the complexity $\mathcal{O}(m^2 nk)$ scales linearly with sample size $n$).

## 3   Hierarchical graph-factorization clustering (HGFC)

As a nice property of the proposed graph factorization, a natural affinity between two clusters $u_p$ and $u_q$ can be computed as

$$p(u_p, u_q) = \sum_{i=1}^{n} \frac{b_{ip} b_{iq}}{d_i} = \left(\mathbf{B}^\top \mathbf{D}^{-1} \mathbf{B}\right)_{pq}, \quad \mathbf{D} = \text{diag}(d_1, \ldots, d_n) \qquad (5)$$

This is similar to Eq. (1), but derived from another way of two-hop transitions $\mathbf{U} \rightarrow \mathbf{V} \rightarrow \mathbf{U}$. Note that the similarity between clusters $p$ and $q$ takes into account a weighted average of contributions from *all* the data (see Fig. 1–(*c*)).

Let $G_0(\mathbf{V}_0, \mathbf{E}_0)$ be the initial graph describing the similarities of totally $m_0 = n$ data points, with adjacency matrix $\mathbf{W}_0$. Based on $G_0$ we can build a bipartite graph $K_1(\mathbf{V}_0, \mathbf{V}_1, \mathbf{F}_1)$, with $m_1 < m_0$ vertices in $\mathbf{V}_1$. A hierarchical clustering method can be motivated from the observation that the cluster similarity in Eq. (5) suggests a new adjacency matrix $\mathbf{W}_1$ for graph $G_1(\mathbf{V}_1, \mathbf{E}_1)$, where $\mathbf{V}_1$ is formed by clusters, and $\mathbf{E}_1$ contains edges connecting these clusters. Then we can group those clusters by constructing another bipartite graph $K_2(\mathbf{V}_1, \mathbf{V}_2, \mathbf{F}_2)$ with $m_2 < m_1$ vertices in $\mathbf{V}_2$, such that $\mathbf{W}_1$ is again factorized as in Eq. (2), and a new graph $G_2(\mathbf{V}_2, \mathbf{E}_2)$ can be built. In principal we can repeat this procedure until we get only one cluster. Algorithm 1 summarizes this algorithm.

---

**Algorithm 1** Hierarchical Graph-Factorization Clustering (HGFC)

---

**Require:** given $n$ data objects and a similarity measure
 1: build the similarity graph $G_0(\mathbf{V}_0, \mathbf{E}_0)$ with adjacency matrix $\mathbf{W}_0$, and let $m_0 = n$
 2: **for** $l = 1, 2, \ldots,$ **do**
 3:     choose $m_l < m_{l-1}$
 4:     factorize $G_{l-1}$ to obtain $K_l(\mathbf{V}_{l-1}, \mathbf{V}_l, \mathbf{F}_l)$ with the adjacency matrix $\mathbf{B}_l$
 5:     build a graph $G_l(\mathbf{V}_l, \mathbf{E}_l)$ with the adjacency matrix $\mathbf{W}_l = \mathbf{B}_l^\top \mathbf{D}_l^{-1} \mathbf{B}_l$, where $\mathbf{D}_l$'s
     diagonal entries are obtained by summation over $\mathbf{B}_l$'s columns
 6: **end for**

---

The algorithm ends up with a hierarchical clustering structure. For level $l$, we can assign data to the obtained $m_l$ clusters via a propagation from the bottom level of clusters. Based on the chain rule of Markov random walks, the soft (i.e., probabilistic) assignment of $v_i \in \mathbf{V}_0$ to cluster $v_p^{(l)} \in \mathbf{V}_l$ is given by

$$p\left(v_p^{(l)}|v_i\right) = \sum_{v^{(l-1)} \in \mathbf{V}_{l-1}} \cdots \sum_{v^{(1)} \in \mathbf{V}_1} p\left(v_p^{(l)}|v^{(l-1)}\right) \cdots p\left(v^{(1)}|v_i\right) = \left(\mathbf{D}_1^{-1}\bar{\mathbf{B}}_l\right)_{ip}, \quad (6)$$

where $\bar{\mathbf{B}}_l = \mathbf{B}_1 \mathbf{D}_2^{-1} \mathbf{B}_2 \mathbf{D}_3^{-1} \mathbf{B}_3 \ldots \mathbf{D}_l^{-1} \mathbf{B}_l$. One can interpret this by deriving an *equivalent* bipartite graph $\bar{K}_l(\mathbf{V}_0, \mathbf{V}_l, \bar{\mathbf{F}}_l)$, and treating $\bar{\mathbf{B}}_l$ as the *equivalent* adjacency matrix attached to the *equivalent* edges $\bar{\mathbf{F}}_l$ connecting data $\mathbf{V}_0$ and clusters $\mathbf{V}_l$.

## 4 Analysis of the proposed algorithms

### 4.1 Flat clustering: statistical modeling of single-hop transitions

In this section we provide some insights to the suggested clustering algorithm, mainly from the perspective of random walks on graphs. Suppose that from a stationary stage of random walks on $G(\mathbf{V}, \mathbf{E})$, one observes $\pi_{ij}$ single-hop transitions between $v_i$ and $v_j$ in a unitary time frame. As an intuition of graph-based view to similarities, if two data points are similar or related, the transitions between them are likely to happen. Thus we connect the observed similarities to the *frequency* of transitions via $w_{ij} \propto \pi_{ij}$. If the observed transitions are i.i.d. sampled from a true distribution $p(v_i, v_j) = (\mathbf{H}\boldsymbol{\Lambda}\mathbf{H}^\top)_{ij}$ where a bipartite graph is behind, then the log likelihood with respect to the observed transitions is

$$\mathcal{L}(\mathbf{H}, \boldsymbol{\Lambda}) = \log \prod_{ij} p(v_i, v_j)^{\pi_{ij}} \propto \sum_{ij} w_{ij} \log(\mathbf{H}\boldsymbol{\Lambda}\mathbf{H}^\top)_{ij}. \quad (7)$$

Then we have the following conclusion

**Proposition 4.1.** *For a weighted undirected graph $G(\mathbf{V}, \mathbf{E})$ and the log likelihood defined in Eq. (7), the following results hold:* (i) *Minimizing the divergence distance $l(\mathbf{W}, \mathbf{H}\boldsymbol{\Lambda}\mathbf{H}^\top)$ is equivalent to maximizing the log likelihood $\mathcal{L}(\mathbf{H}, \boldsymbol{\Lambda})$;* (ii) *Updates Eq. (3) and Eq. (4) correspond to a standard EM algorithm for maximizing $\mathcal{L}(\mathbf{H}, \boldsymbol{\Lambda})$.*

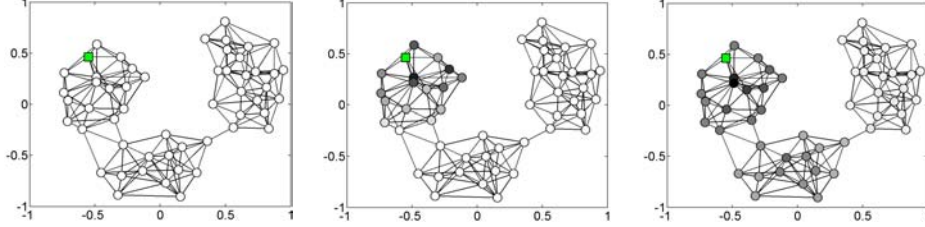

Figure 2: The similarities of vertices to a fixed vertex (marked in the left panel) on a 6-nearest-neighbor graph, respectively induced by clustering level $l = 2$ (the middle panel) and $l = 6$ (the right panel). A darker color means a higher similarity.

### 4.2 Hierarchical clustering: statistical modeling of multi-hop transitions

The adjacency matrix $\mathbf{W}_0$ of $G_0(\mathbf{V}_0, \mathbf{E}_0)$ only models one-hop transitions that follow direct links from vertices to their neighbors. However, the random walk is a process of diffusion on the graph. Within a relatively longer period, a walker starting from a vertex has the chance to reach vertices faraway through *multi-hop* transitions. Obviously, multi-hop transitions induce a slowly decaying similarity function on the graph. Based on the chain rule of Markov process, the equivalent adjacency matrix for $t$-hop transitions is

$$\mathbf{A}_t = \mathbf{W}_0(\mathbf{D}_0^{-1}\mathbf{W}_0)^{t-1} = \mathbf{A}_{t-1}\mathbf{D}_0^{-1}\mathbf{W}_0. \tag{8}$$

Generally speaking, a slowly decaying similarity function on the similarity graph captures a global affinity structure of data manifolds, while a rapidly decaying similarity function only tells the local affinity structure. The following proposition states that in the suggested HGFC, a higher-level clustering implicitly employs a more global similarity measure caused by multi-hop Markov random walks:

**Proposition 4.2.** *For a given hierarchical clustering structure that starts from a bottom graph $G_0(\mathbf{V}_0, \mathbf{E}_0)$ to a higher level $G_k(\mathbf{V}_k, \mathbf{E}_k)$, the vertices $\mathbf{V}_l$ at level $0 < l \leq k$ induces an equivalent adjacency matrix of $\mathbf{V}_0$, which is $\mathbf{A}_t$ with $t = 2^{l-1}$ as defined in Eq. (8).*

Therefore the presented hierarchical clustering algorithm HGFC applies different sizes of time windows to examine random walks, and derives different scales of similarity measures to expose the local and global clustering structures of data manifolds. Fig. 2 illustrates the employed similarities of vertices to a fixed vertex in clustering levels $l = 2$ and 6, which corresponds to time periods $t = 2$ and 32. It can be seen that for a short period $t = 2$, the similarity is very local and helps to uncover low-level clusters, while in a longer period $t = 32$ the similarity function is rather global.

## 5   Empirical study

We apply HGFC on USPS handwritten digits and Newsgroup text data. For USPS data we use the images of digits 1, 2, 3 and 4, with respectively 1269, 929, 824 and 852 images per class. Each image is represented as a 256-dimension vector. The text data contain totally 3970 documents covering 4 categories, autos, motorcycles, baseball, and hockey. Each document is represented by an 8014-dimension TFIDF feature vector. Our method employs a 10-nearest-neighbor graph, with the similarity measure RBF for USPS and cosine for Newsgroup. We perform 4-level HGFC, and set the cluster number, respectively from bottom to top, to be 100, 20, 10 and 4 for both data sets.

We compare HGFC with two popular agglomerative hierarchical clustering algorithms, single link and complete link (e.g., [3]). Both methods merge two closest clusters at each step.

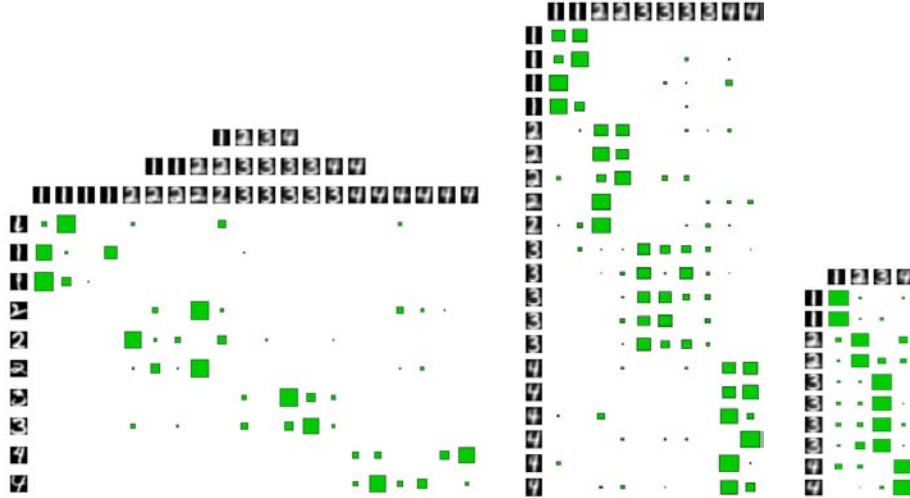

Figure 3: Visualization of HGFC for USPS data set. Left: mean images of the top 3 clustering levels, along with a Hinton graph representing the soft (probabilistic) assignments of randomly chosen 10 digits (shown on the left) to the top 3rd level clusters; Middle: a Hinton graph showing the soft cluster assignments from top 3rd level to top 2nd level; Right: a Hinton graph showing the soft assignments from top 2nd level to top 1st level.

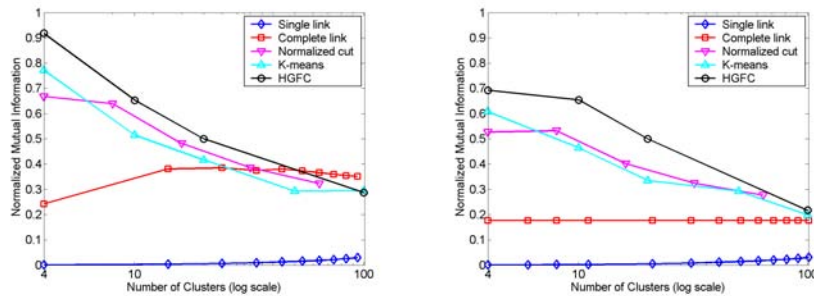

Figure 4: Comparison of clustering methods on USPS (left) and Newsgroup (right), evaluated by normalized mutual information (NMI). Higher values indicate better qualities.

Single link defines the cluster distance to be the smallest point-wise distance between two clusters, while complete link uses the largest one. A third compared method is normalized cut [5], which partitions data into two clusters. We apply the algorithm recursively to produce a top-down hierarchy of 2, 4, 8, 16, 32 and 64 clusters. We also compare with the k-means algorithm, $k = 4, 10, 20$ and $100$.

Before showing the comparison, we visualize a part of clustering results for USPS data in Fig. 3. On top of the left figure, we show the top three levels of the hierarchy with respectively 4, 10 and 20 clusters, where each cluster is represented by its mean image via an average over all the images weighted by their posterior probabilities of belonging to this cluster. Then 10 randomly sampled digits with soft cluster assignments to the top 3rd level clusters are illustrated with a Hinton graph. The middle and right figures in Fig. 3 show the assignments between clusters across the hierarchy. The clear diagonal block structure in all the Hinton graphs indicates a very meaningful cluster hierarchy.

| | Normalized cut | | | | HGFC | | | | K-means | | | |
|---|---|---|---|---|---|---|---|---|---|---|---|---|
| "1" | 635 | 630 | 1 | 3 | **1254** | 3 | 8 | 4 | 1265 | 1 | 0 | 3 |
| "2" | 2 | 4 | 744 | 179 | 1 | **886** | 33 | 9 | 17 | 720 | 95 | 97 |
| "3" | 2 | 1 | 817 | 4 | 1 | 4 | **816** | 3 | 10 | 9 | 796 | 9 |
| "4" | 10 | 6 | 1 | 835 | 4 | 8 | 2 | **838** | 58 | 20 | 0 | 774 |

Table 1: Confusion matrices of clustering results, 4 clusters, USPS data. In each confusion matrix, rows correspond true classes and columns correspond the found clusters.

| | Normalized cut | | | | HGFC | | | | K-means | | | |
|---|---|---|---|---|---|---|---|---|---|---|---|---|
| autos | 858 | 98 | 30 | 2 | **772** | 182 | 13 | 21 | 977 | 7 | 4 | 0 |
| motor. | 79 | 893 | 16 | 5 | 42 | **934** | 5 | 12 | 985 | 3 | 5 | 0 |
| baseball | 44 | 33 | 875 | 40 | 15 | 33 | **843** | 101 | 39 | 835 | 114 | 4 |
| hockey | 11 | 8 | 893 | 85 | 7 | 21 | 11 | **958** | 16 | 4 | 900 | 77 |

Table 2: Confusion matrices of clustering results, 4 clusters, Newsgroup data. In each confusion matrix, rows correspond true classes and columns correspond the found clusters.

We compare the clustering methods by evaluating the normalized mutual information (NMI) in Fig. 4. It is defined to be the mutual information between clusters and true classes, normalized by the maximum of marginal entropies. Moreover, in order to more directly assess the clustering quality, we also illustrate the confusion matrices in Table 1 and Table 2, in the case of producing 4 clusters. We drop out the confusion matrices of single link and complete link in the tables, for saving spaces and also due to their clearly poor performance compared with others.

The results show that single link performs poorly, as it greedily merges nearby data and tends to form a big cluster with some outliers. Complete link is more balanced but unsatisfactory either. For the Newsgroup data it even gets stuck at the 3601-th merge because all the similarities between clusters are 0. Top-down hierarchical normalized cut obtains reasonable results, but sometimes cannot split one big cluster (see the tables). The confusion matrices indicates that k-means does well for digit images but relatively worse for high-dimension textual data. In contrast, Fig. 4 shows that HGFC gives significantly higher NMI values than competitors on both tasks. It also produces confusion matrices with clear diagonal structures (see tables 1 and 2), which indicates a very good clustering quality.

# 6 Conclusion and Future Work

In this paper we have proposed a probabilistic graph partition method for clustering data objects based on their pairwise similarities. A novel hierarchical clustering algorithm HGFC has been derived, where a higher level in HGFC corresponds to a statistical model of random walk transitions in a longer period, giving rise to a more global clustering structure. Experiments show very encouraging results.

In this paper we have empirically specified the number of clusters in each level. In the near future we plan to investigate effective methods to automatically determine it. Another direction is hierarchical clustering on directed graphs, as well as its applications in web mining.

# Appendix

*Proof of Theorem 2.1.* We first notice that $\sum_p \lambda_p = \sum_{ij} w_{ij}$ under constraints $\sum_i h_{ip} = 1$. Therefore we can normalize $\mathbf{W}$ by $\sum_{ij} w_{ij}$ and after convergence multiply all $\lambda_p$ by this quantity to get the solution. Under this assumption we are maximizing $\mathcal{L}(\mathbf{H}, \mathbf{\Lambda}) = \sum_{ij} w_{ij} \log(\mathbf{H}\mathbf{\Lambda}\mathbf{H}^\top)_{ij}$ with an extra constraint $\sum_p \lambda_p = 1$. We first fix $\lambda_p$ and show update Eq. (3) will not decrease $\mathcal{L}(\mathbf{H}) \equiv \mathcal{L}(\mathbf{H}, \mathbf{\Lambda})$. We prove this by constructing an auxiliary function $f(\mathbf{H}, \mathbf{H}^*)$ such that $f(\mathbf{H}, \mathbf{H}^*) \leq \mathcal{L}(\mathbf{H})$ and $f(\mathbf{H}, \mathbf{H}) = \mathcal{L}(\mathbf{H})$. Then we know the update $\mathbf{H}^{t+1} = \arg\max_{\mathbf{H}} f(\mathbf{H}, \mathbf{H}^t)$ will not decrease $\mathcal{L}(\mathbf{H})$ since $\mathcal{L}(\mathbf{H}^{t+1}) \geq f(\mathbf{H}^{t+1}, \mathbf{H}^t) \geq f(\mathbf{H}^t, \mathbf{H}^t) = \mathcal{L}(\mathbf{H}^t)$. Define $f(\mathbf{H}, \mathbf{H}^*) = \sum_{ij} w_{ij} \sum_p \frac{h_{ip}^* \lambda_p h_{jp}^*}{\sum_l h_{il}^* \lambda_l h_{jl}^*} \left( \log h_{ip} \lambda_p h_{jp} - \log \frac{h_{ip}^* \lambda_p h_{jp}^*}{\sum_l h_{il}^* \lambda_l h_{jl}^*} \right)$. $f(\mathbf{H}, \mathbf{H}) = \mathcal{L}(\mathbf{H})$ can be easily verified, and $f(\mathbf{H}, \mathbf{H}^*) \leq \mathcal{L}(\mathbf{H})$ also follows if we use concavity of log function. Then it is straightforward to verify Eq. (3) by setting the derivative of $f$ with respect to $h_{ip}$ to be zero. The normalization is due to the constraints and can be formally derived from this procedure with a Lagrange formalism. Similarly we can define an auxiliary function for $\mathbf{\Lambda}$ with $\mathbf{H}$ fixed, and verify Eq. (4). $\qquad\square$

*Proof of Proposition 4.1.* (i) follows directly from the proof of Theorem 2.1. To prove (ii) we take $u_p$ as the missing data and follow the standard way to derive the EM algorithm. In the E-step we estimate the *a posteriori* probability of taking $u_p$ for pair $(v_i, v_j)$ using Bayes' rule: $\hat{p}(u_p|v_i, v_j) \propto p(v_i|u_p)p(v_j|u_p)p(u_p)$. And then in the M-step we maximize the "complete" data likelihood $\hat{\mathcal{L}}(G) = \sum_{ij} w_{ij} \sum_p \hat{p}(u_p|v_i, v_j) \log p(v_i|u_p)p(v_j|u_p)p(u_p)$ with respect to model parameters $h_{ip} = p(v_i|u_p)$ and $\lambda_p = p(u_p)$, with constraints $\sum_i h_{ip} = 1$ and $\sum_p \lambda_p = 1$. By setting the corresponding derivatives to zero we obtain $h_{ip} \propto \sum_j w_{ij} \hat{p}(u_p|v_i, v_j)$ and $\lambda_p \propto \sum_{ij} w_{ij} \hat{p}(u_p|v_i, v_j)$. It is easy to check that they are equivalent to updates Eq. (3) and Eq. (4) respectively. $\qquad\square$

*Proof of Proposition 4.2.* We give a brief proof. Suppose that at level $l$ the data-cluster relationship is described by $\bar{K}_l(\mathbf{V}_0, \mathbf{V}_l, \bar{\mathbf{F}}_l)$ (see Eq. (6)) with adjacency matrix $\bar{\mathbf{B}}_l$, degrees $\mathbf{D}_0$ for $\mathbf{V}_0$, and degrees $\mathbf{\Lambda}_l$ for $\mathbf{V}_l$. In this case the induced adjacency matrix of $\mathbf{V}_0$ is $\bar{\mathbf{W}}_l = \bar{\mathbf{B}}_l \mathbf{\Lambda}_l^{-1} \bar{\mathbf{B}}_l^\top$, and the adjacency matrix of $\mathbf{V}_l$ is $\mathbf{W}_l = \bar{\mathbf{B}}_l^\top \mathbf{D}_0^{-1} \bar{\mathbf{B}}_l$. Let $K_l(\mathbf{V}_l, \mathbf{V}_{l+1}, \mathbf{F}_{l+1})$ be the bipartite graph connecting $\mathbf{V}_l$ and $\mathbf{V}_{l+1}$, with the adjacency $\mathbf{B}_{l+1}$ and degrees $\mathbf{\Lambda}_{l+1}$ for $\mathbf{V}_{l+1}$. Then the adjacency matrix of $\mathbf{V}_0$ induced by level $l + 1$ is $\bar{\mathbf{W}}_{l+1} = \bar{\mathbf{B}}_l \mathbf{\Lambda}_l^{-1} \mathbf{B}_{l+1} \mathbf{\Lambda}_{l+1}^{-1} \mathbf{B}_{l+1}^\top \mathbf{\Lambda}_l^{-1} \bar{\mathbf{B}}_l^\top = \bar{\mathbf{W}}_l \mathbf{D}_0^{-1} \bar{\mathbf{W}}_l$, where relations $\mathbf{B}_{l+1} \mathbf{\Lambda}_{l+1}^{-1} \mathbf{B}_{l+1}^\top = \bar{\mathbf{B}}_l^\top \mathbf{D}_0^{-1} \bar{\mathbf{B}}_l$ and $\bar{\mathbf{W}}_l = \mathbf{B}_l \mathbf{\Lambda}_l^{-1} \mathbf{B}_l^\top$ are applied. Given the initial condition from the bottom level $\bar{\mathbf{W}}_1 = \mathbf{W}_0$, it is not difficult to obtain $\bar{\mathbf{W}}_l = \mathbf{A}_t$ with $t = 2^{l-1}$. $\qquad\square$

# References

[1] J. Goldberger and S. Roweis. Hierarchical clustering of a mixture model. In L.K. Saul, Y. Weiss, and L. Bottou, editors, *Neural Information Processing Systems 17 (NIPS*04)*, pages 505–512, 2005.

[2] K.A. Heller and Z. Ghahramani. Bayesian hierarchical clustering. In *Proceedings of the 22nd International Conference on Machine Learning*, pages 297–304, 2005.

[3] S. D. Kamvar, D. Klein, and C. D. Manning. Interpreting and extending classical agglomerative clustering algorithms using a model-based approach. In *Proceedings of the 19th International Conference on Machine Learning*, pages 283–290, 2002.

[4] Daniel D. Lee and H. Sebastian Seung. Algorithms for non-negative matrix factorization. In T. K. Leen, T. G. Dietterich, and V. Tresp, editors, *Advances in Neural Information Processing Systems 13 (NIPS*00)*, pages 556–562, 2001.

[5] Jianbo Shi and Jitendra Malik. Normalized cuts and image segmentation. *IEEE Transactions on Pattern Analysis and Machine Intelligence*, 22(8):888–905, 2000.

[6] D. Zhou, B. Schölkopf, and T. Hofmann. Semi-supervised learning on directed graphs. In L.K. Saul, Y. Weiss, and L. Bottou, editors, *Advances in Neural Information Processing Systems 17 (NIPS*04)*, pages 1633–1640, 2005.
